# New Adaptive Algorithms for Online Classification

**Francesco Orabona**
DSI
Università degli Studi di Milano
Milano, 20135 Italy
orabona@dsi.unimi.it

**Koby Crammer**
Department of Electrical Enginering
The Technion
Haifa, 32000 Israel
koby@ee.technion.ac.il

## Abstract

We propose a general framework to online learning for classification problems with time-varying potential functions in the adversarial setting. This framework allows to design and prove relative mistake bounds for any generic loss function. The mistake bounds can be specialized for the hinge loss, allowing to recover and improve the bounds of known online classification algorithms. By optimizing the general bound we derive a new online classification algorithm, called NAROW, that hybridly uses adaptive- and fixed- second order information. We analyze the properties of the algorithm and illustrate its performance using synthetic dataset.

## 1 Introduction

Linear discriminative online algorithms have been shown to perform very well on binary and multiclass labeling problems [10, 6, 14, 3]. These algorithms work in rounds, where at each round a new instance is given and the algorithm makes a prediction. After the true class of the instance is revealed, the learning algorithm updates its internal hypothesis. Often, such update is taking place only on rounds where the online algorithm makes a prediction mistake or when the confidence in the prediction is not sufficient. The aim of the classifier is to minimize the cumulative loss it suffers due to its prediction, such as the total number of mistakes.

Until few years ago, most of these algorithms were using only first-order information of the input features. Recently [1, 8, 4, 12, 5, 9], researchers proposed to improve online learning algorithms by incorporating second order information. Specifically, the Second-Order-Perceptron (SOP) proposed by Cesa-Bianchi et al. [1] builds on the famous Perceptron algorithm with an additional data-dependent time-varying "whitening" step. Confidence weighted learning (CW) [8, 4] and the adaptive regularization of weights algorithm (AROW) [5] are motivated from an alternative view: maintaining confidence in the weights of the linear models maintained by the algorithm. Both CW and AROW use the input data to modify the weights as well and the confidence in them. CW and AROW are motivated from the specific properties of natural-language-precessing (NLP) data and indeed were shown to perform very well in practice, and on NLP problems in particular. However, the theoretical foundations of this empirical success were not known, especially when using only the diagonal elements of the second order information matrix. Filling this gap is one contribution of this paper.

In this paper we extend and generalizes the framework for deriving algorithms and analyzing them through a potential function [2]. Our framework contains as a special case the second order Perceptron and a (variant of) AROW. While it can also be used to derive new algorithms based on other loss functions.

For carefully designed algorithms, it is possible to bound the cumulative loss on any sequence of samples, even adversarially chosen [2]. In particular, many of the recent analyses are based on the online convex optimization framework, that focuses on minimizing the sum of convex functions.

Two common view-points for online convex optimization are of regularization [15] or primal-dual progress [16, 17, 13]. Recently new bounds have been proposed for time-varying regularizations in [18, 9], focusing on the general case of regression problems. The proof technique derived from our framework extends the work of Kakade et al. [13] to support time varying potential functions. We also show how the use of widely used classification losses, as the hinge loss, allows us to derive new powerful mistake bounds superior to existing bounds. Moreover the framework introduced supports the design of aggressive algorithms, i.e. algorithms that update their hypothesis not only when they make a prediction mistake.

Finally, current second order algorithms suffer from a common problem. All these algorithms maintain the cumulative second-moment of the input features, and its inverse, qualitatively speaking, is used as a learning rate. Thus, if there is a single feature with large second-moment in the prefix of the input sequence, its effective learning rate would drop to a relatively low value, and the learning algorithm will take more time to update its value. When the instances are ordered such that the value of this feature seems to be correlated with the target label, such algorithms will set the value of weight corresponding to this feature to a wrong value *and* will decrease its associated learning rate to a low value. This combination makes it hard to recover from the wrong value set to the weight associated with this feature. Our final contribution is a new algorithm that adapts the way the second order information is used. We call this algorithm Narrow Adaptive Regularization Of Weights (NAROW). Intuitively, it interpolates its update rule from adaptive-second-order-information to fixed-second-order-information, to have a narrower decrease of the learning rate for common appearing features. We derive a bound for this algorithm and illustrate its properties using synthetic data simulations.

## 2 Online Learning for Classification

We work in the online binary classification scenario where learning algorithms work in rounds. At each round $t$, an instance $\boldsymbol{x}_t \in \mathbb{R}^d$ is presented to the algorithm, which then predicts a label $\hat{y}_t \in \{-1, +1\}$. Then, the correct label $y_t$ is revealed, and the algorithm may modify its hypothesis. The aim of the online learning algorithm is to make as few mistakes as possible (on any sequence of samples/labels $\{(\boldsymbol{x}_t, y_t)\}_{t=1}^T$). In this paper we focus on linear prediction functions of the form $\hat{y}_t = \text{sign}(\boldsymbol{w}_t^\top \boldsymbol{x}_t)$.

We strive to design online learning algorithms for which it is possible to prove a relative mistakes bound or a loss bound. Typical such analysis bounds the cumulative loss the algorithm suffers, $\sum_{t=1}^T \ell(\boldsymbol{w}_t, \boldsymbol{x}_t, y_t)$, with the cumulative loss of any classifier $\boldsymbol{u}$ plus an additional penalty called the regret, $R(\boldsymbol{u}) + \sum_{t=1}^T \ell(\boldsymbol{u}, \boldsymbol{x}_t, y_t)$. Given that we focus on classification, we are more interested in relative mistakes bound, where we bound the number of mistakes of the learner with $R(\boldsymbol{u}) + \sum_{t=1}^T \ell(\boldsymbol{u}, \boldsymbol{x}_t, y_t)$. Since the classifier $\boldsymbol{u}$ is arbitrary, we can choose, in particular, the best classifier that can be found in hindsight given all the samples. Often $R(\cdot)$ depends on a function measuring the complexity of $\boldsymbol{u}$ and the number of samples $T$, and $\ell$ is a non-negative loss function. Usually $\ell$ is chosen to be a convex upper bound of the 0/1 loss. We will also denote by $\ell_t(\boldsymbol{u}) = \ell(\boldsymbol{u}, \boldsymbol{x}_t, y_t)$.

In the following we denote by $\mathcal{M}$ to be the set of round indexes for which the algorithm performed a mistake. We assume that the algorithm always update if it rules in such events. Similarly, we denote by $\mathcal{U}$ the set of the margin error rounds, that is, rounds in which the algorithm updates its hypothesis and the prediction is correct, but the loss $\ell_t(\boldsymbol{w}_t)$ is different from zero. Their cardinality will be indicated with $M$ and $U$ respectively. Formally, $\mathcal{M} = \{t : \text{sign}(\boldsymbol{w}_t^\top \boldsymbol{x}_t) \neq y_t \ \& \ \boldsymbol{w}_t \neq \boldsymbol{w}_{t+1}\}$, and $\mathcal{U} = \{t : \text{sign}(\boldsymbol{w}_t^\top \boldsymbol{x}_t) = y_t \ \& \ \boldsymbol{w}_t \neq \boldsymbol{w}_{t+1}\}$. An algorithm that updates its hypothesis only on mistake rounds is called *conservative* (e.g. [3]). Following previous naming convention [3], we call *aggressive* an algorithm that updates is rule on rounds for which the loss $\ell_t(\boldsymbol{w}_t)$ is different from zero, even if its prediction was correct.

We define now few basic concepts from convex analysis that will be used in the paper. Given a convex function $f : \mathbb{X} \to \mathbb{R}$, its sub-gradient $\partial f(\boldsymbol{v})$ at $\boldsymbol{v}$ satisfies: $\forall \boldsymbol{u} \in \mathbb{X}, f(\boldsymbol{u}) - f(\boldsymbol{v}) \geq (\boldsymbol{u} - \boldsymbol{v}) \cdot \partial f(\boldsymbol{v})$. The Fenchel conjugate of $f$, $f^* : S \to \mathbb{R}$, is defined by $f^*(\boldsymbol{u}) = \sup_{\boldsymbol{v} \in S} (\boldsymbol{v} \cdot \boldsymbol{u} - f(\boldsymbol{v}))$. A differentiable function $f : \mathbb{X} \to \mathbb{R}$ is $\beta$-strongly convex w.r.t. a norm $\|\cdot\|$ if for any $\boldsymbol{u}, \boldsymbol{v} \in S$ and $\alpha \in (0, 1)$, $h(\alpha \boldsymbol{u} + (1-\alpha)\boldsymbol{v}) \leq \alpha f(\boldsymbol{u}) + (1-\alpha)f(\boldsymbol{v}) - \frac{\beta}{2}\alpha(1-\alpha)\|\boldsymbol{u} - \boldsymbol{v}\|^2$. Strong convexity turns out to be a key property to design online learning algorithms.

# 3 General Algorithm and Analysis

We now introduce a general framework to design online learning algorithms and a general lemma which serves as a general tool to prove their relative regret bounds. Our algorithm builds on previous algorithms for online convex programming with a one significant difference. Instead of using a fixed link function as first order algorithms, we allow a sequence of link functions $f_t(\cdot)$, one for each time $t$. In a nutshell, the algorithm maintains a weight vector $\boldsymbol{\theta}_t$. Given a new examples it uses the current link function $f_t$ to compute a prediction weight vector $\boldsymbol{w}_t$. After the target label is received it sets the new weight $\boldsymbol{\theta}_{t+1}$ to be the sum of $\boldsymbol{\theta}_t$ and minus the gradient of the loss at $\boldsymbol{w}_t$. The algorithm is summarized in Fig. 1.

The following lemma is a generalization of Corollary 7 in [13] and Corollary 3 in [9], for online learning. All the proofs can be found in the Appendix.

**Lemma 1.** *Let $f_t, t = 1, \ldots, T$ be $\beta_t$-strongly convex functions with respect to the norms $\| \cdot \|_{f_1}, \ldots, \| \cdot \|_{f_T}$ over a set $S$ and let $\| \cdot \|_{f_i^*}$ be the respective dual norms. Let $f_0(\mathbf{0}) = 0$, and $\boldsymbol{x}_1, \ldots, \boldsymbol{x}_T$ be an arbitrary sequence of vectors in $\mathbb{R}^d$. Assume that algorithm in Fig. 1 is run on this sequence with the functions $f_i$. Then, for any $\boldsymbol{u} \in S$, and any $\lambda > 0$ we have*

$$\sum_{t=1}^{T} \eta_t \boldsymbol{z}_t^\top \left( \frac{1}{\lambda} \boldsymbol{w}_t - \boldsymbol{u} \right) \leq \frac{f_T(\lambda \boldsymbol{u})}{\lambda} + \sum_{t=1}^{T} \left( \frac{\eta_t^2 \|\boldsymbol{z}_t\|_{f_t^*}^2}{2\lambda \beta_t} + \frac{1}{\lambda} (f_t^*(\boldsymbol{\theta}_t) - f_{t-1}^*(\boldsymbol{\theta}_t)) \right) .$$

This Lemma can appear difficult to interpret, but we now show that it is straightforward to use the lemma to recover known bounds of different online learning algorithms. In particular we can state the following Corollary that holds for any convex loss $\ell$ that upper bounds the 0/1 loss.

<div style="display: flex">

1: **Input:** A series of strongly convex functions $f_1, \ldots, f_T$.
2: **Initialize:** $\boldsymbol{\theta}_1 = \mathbf{0}$
3: **for** $t = 1, 2, \ldots, T$ **do**
4:     Receive $\boldsymbol{x}_t$
5:     Set $\boldsymbol{w}_t = \nabla f_t^*(\theta_t)$
6:     Predict $\hat{y}_t = sign(\boldsymbol{w}_t^\top \boldsymbol{x}_t)$
7:     Receive $y_t$
8:     **if** $\ell_t(\boldsymbol{w}_t) > 0$ **then**
9:         $\boldsymbol{z}_t = \partial \ell_t(\boldsymbol{w}_t)$
10:         $\boldsymbol{\theta}_{t+1} = \boldsymbol{\theta}_t - \eta_t \boldsymbol{z}_t$
11:     **else**
12:         $\boldsymbol{\theta}_{t+1} = \boldsymbol{\theta}_t$
13:     **end if**
14: **end for**

Figure 1: Prediction algorithm

</div>

**Corollary 1.** *Define $B = \sum_{t=1}^{T} (f_t^*(\boldsymbol{\theta}_t) - f_{t-1}^*(\boldsymbol{\theta}_t))$. Under the hypothesis of Lemma 1, if $\ell$ is convex and it upper bounds the 0/1 loss, and $\eta_t = \eta$, then for any $\boldsymbol{u} \in S$ the algorithm in Fig. 1 has the following bound on the maximum number of mistakes $M$,*

$$M \leq \sum_{t=1}^{T} \ell_t(\boldsymbol{u}) + \frac{f_T(\boldsymbol{u})}{\eta} + \eta \sum_{t=1}^{T} \frac{\|\boldsymbol{z}_t\|_{f_t^*}^2}{2\beta_t} + \frac{B}{\eta} . \quad (1)$$

*Moreover if $f_t(\boldsymbol{x}) \leq f_{t+1}(\boldsymbol{x}), \forall \boldsymbol{x} \in S, t = 0, \ldots, T-1$ then $B \leq 0$.*

A similar bound has been recently presented in [9] as a regret bound. Yet, there are two differences. First, our analysis bounds the number of mistakes, a more natural quantity in classification setting, rather than of a general loss function. Second, we retain the additional term $B$ which may be negative, and thus possibly provide a better bound. Moreover, to choose the optimal tuning of $\eta$ we should know quantities that are unknown to the learner. We could use adaptive regularization methods, as the one proposed in [16, 18], but in this way we would lose the possibility to prove mistake bounds for second order algorithms, like the ones in [1, 5]. In the next Section we show how to obtain bounds with an automatic tuning, using additional assumption-ion on the loss function.

## 3.1 Better bounds for linear losses

The hinge loss, $\ell(\boldsymbol{u}, \boldsymbol{x}_t, y_t) = \max(1 - y_t \boldsymbol{u}^\top \boldsymbol{x}_t, 0)$, is a very popular evaluation metric in classification. It has been used, for example, in Support Vector Machines [7] as well as in many online learning algorithms [3]. It has also been extended to the multiclass case [3]. Often mistake bounds are expressed in terms of the hinge loss. One reason is that it is a tighter upper bound of the 0/1 loss compared to other losses, as the squared hinge loss. However, this loss is particularly interesting for us, because it allows an automatic tuning of the bound in (1). In particular it is easy to verify that it satisfies the following condition

$$\ell(\boldsymbol{u}, \boldsymbol{x}_t, y_t) \geq 1 + \boldsymbol{u}^\top \partial \ell_t(\boldsymbol{w}_t), \ \forall \boldsymbol{u} \in S, \boldsymbol{w}_t : \ell_t(\boldsymbol{w}_t) > 0 . \quad (2)$$

Thanks to this condition we can state the following Corollary for any loss satisfying (2).

**Corollary 2.** *Under the hypothesis of Lemma 1, if $f_T(\lambda \boldsymbol{u}) \leq \lambda^2 f_T(\boldsymbol{u})$, and $\ell$ satisfies (2), then for any $\boldsymbol{u} \in S$, and any $\lambda > 0$ we have*

$$\sum_{t \in \mathcal{M} \cup \mathcal{U}} \eta_t \leq L + \lambda f_T(\boldsymbol{u}) + \frac{1}{\lambda}\left( B + \sum_{t \in \mathcal{M} \cup \mathcal{U}} \left( \frac{\eta_t^2}{2\beta_t} \|\boldsymbol{z}_t\|_{f_t^*}^2 - \eta_t \boldsymbol{w}_t^\top \boldsymbol{z}_t \right) \right),$$

*where $L = \sum_{t \in \mathcal{M} \cup \mathcal{U}} \eta_t \ell_t(\boldsymbol{u})$, and $B = \sum_{t=1}^{T}(f_t^*(\boldsymbol{\theta}_t) - f_{t-1}^*(\boldsymbol{\theta}_t))$. In particular, choosing the optimal $\lambda$, we obtain*

$$\sum_{t \in \mathcal{M} \cup \mathcal{U}} \eta_t \leq L + \sqrt{2 f_T(\boldsymbol{u})} \sqrt{2B + \sum_{t \in \mathcal{M} \cup \mathcal{U}} \left( \frac{\eta_t^2}{\beta_t} \|\boldsymbol{z}_t\|_{f_t^*}^2 - 2\eta_t \boldsymbol{w}_t^\top \boldsymbol{z}_t \right)}. \tag{3}$$

The intuition and motivation behind this Corollary is that a classification algorithm should be independent of the particular scaling of the hyperplane. In other words, $\boldsymbol{w}_t$ and $\alpha \boldsymbol{w}_t$ (with $\alpha > 0$) make exactly the same predictions, because only the sign of the prediction matters. Exactly this independence in a scale factor allows us to improve the mistake bound (1) to the bound of (3). Hence, when (2) holds, the update of the algorithm becomes somehow independent from the scale factor, and we have the better bound. Finally, note that when the hinge loss is used, the vector $\boldsymbol{\theta}_t$ is updated as in an aggressive version of the Perceptron algorithm, with a possible variable learning rate.

## 4 New Bounds for Existing Algorithms

We now show the versatility of our framework, proving better bounds for some known first order and second order algorithms.

### 4.1 An Aggressive p-norm Algorithm

We can use the algorithm in Fig. 1 to obtain an aggressive version of the p-norm algorithm [11]. Set $f_t(\boldsymbol{u}) = \frac{1}{2(q-1)}\|\boldsymbol{u}\|_q^2$, that is 1-strongly convex w.r.t. the norm $\|\cdot\|_q$. The dual norm of $\|\cdot\|_q$ is $\|\cdot\|_p$, where $1/p + 1/q = 1$. Moreover set $\eta_t = 1$ in mistake error rounds, so using the second bound of Corollary 2, and defining $R$ such that $\|\boldsymbol{x}_t\|_p^2 \leq R^2$, we have

$$M \leq L + \sqrt{\frac{\|\boldsymbol{u}\|_q^2}{q-1}} \sqrt{\sum_{t \in \mathcal{M} \cup \mathcal{U}} \left( \eta_t^2 \|\boldsymbol{x}_t\|_p^2 + 2\eta_t y_t \boldsymbol{w}_t^\top \boldsymbol{x}_t \right)} - \sum_{t \in \mathcal{U}} \eta_t$$

$$\leq L + \sqrt{\frac{\|\boldsymbol{u}\|_q^2}{q-1}} \sqrt{M R^2 + \sum_{t \in \mathcal{U}} \left( \eta_t^2 \|\boldsymbol{x}_t\|_p^2 + 2\eta_t y_t \boldsymbol{w}_t^\top \boldsymbol{x}_t \right)} - \sum_{t \in \mathcal{U}} \eta_t.$$

Solving for $M$ we have

$$M \leq L + \frac{1}{2(q-1)}\|\boldsymbol{u}\|_q^2 R^2 + R\frac{\|\boldsymbol{u}\|_q}{\sqrt{q-1}}\sqrt{\frac{1}{4(q-1)}\|\boldsymbol{u}\|_q^2 R^2 + L + D} - \sum_{t \in \mathcal{U}} \eta_t, \tag{4}$$

where $L = \sum_{t \in \mathcal{M} \cup \mathcal{U}} \eta_t \ell_t(\boldsymbol{u})$, and $D = \sum_{t \in \mathcal{U}} \left( \frac{\eta_t^2 \|\boldsymbol{x}_t\|_p^2 + 2\eta_t y_t \boldsymbol{w}_t^\top \boldsymbol{x}_t}{R^2} - \eta_t \right)$. We have still the freedom to set $\eta_t$ in margin error rounds. If we set $\eta_t = 0$, the algorithm of Fig. 1 becomes the p-norm algorithm and we recover its best bound [11]. However if $0 \leq \eta_t \leq \min\left( \frac{R^2 - 2y_t \boldsymbol{w}_t^\top \boldsymbol{x}_t}{\|\boldsymbol{x}_t\|_p^2}, 1 \right)$ we have that $D$ is negative, and $L \leq \sum_{t \in \mathcal{M} \cup \mathcal{U}} \ell_t(\boldsymbol{u})$. Hence the aggressive updates gives us a better bound, thanks to last term that is subtracted to the bound.

In the particular case of $p = q = 2$ we recover the Perceptron algorithm. In particular the minimum of $D$, under the constraint $\eta_t \leq 1$, can be found setting $\eta_t = \min\left( \frac{R^2/2 - y_t \boldsymbol{w}_t^\top \boldsymbol{x}_t}{\|\boldsymbol{x}_t\|^2}, 1 \right)$. If $R$ is equal to $\sqrt{2}$, we recover the PA-I update rule, when $C = 1$. However note that the mistake bound in (4) is better than the one proved for PA-I in [3] and the ones in [16]. Hence the bound (4) provides the first theoretical justification to the good performance of the PA-I, and it can be seen as a general evidence supporting the aggressive updates versus the conservative ones.

## 4.2 Second Order Algorithms

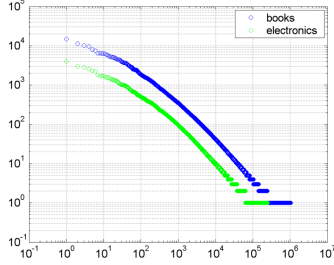

Figure 2: NLP Data: the number of words vs. the word-rank on two sentiment data sets.

We show now how to derive in a simple way the bound of the SOP [1] and the one of AROW [5]. Set $f_t(\boldsymbol{x}) = \frac{1}{2}\boldsymbol{x}^\top A_t \boldsymbol{x}$, where $A_t = A_{t-1} + \frac{\boldsymbol{x}_t \boldsymbol{x}_t^\top}{r}$, $r > 0$ and $A_0 = I$. The functions $f_t$ are 1-strongly convex w.r.t. the norms $\|\boldsymbol{x}\|_{f_t}^2 = \boldsymbol{x}^\top A_t \boldsymbol{x}$. The dual functions of $f_t(\boldsymbol{x})$, $f_t^*(\boldsymbol{x})$, are equal to $\frac{1}{2}\boldsymbol{x}^\top A_t^{-1}\boldsymbol{x}$, while $\|\boldsymbol{x}\|_{f_t^*}^2$ is $\boldsymbol{x}^\top A_t^{-1}\boldsymbol{x}$. Denote by $\chi_t = \boldsymbol{x}_t^\top A_{t-1}^{-1} \boldsymbol{x}_t$ and $m_t = y_t \boldsymbol{x}_t^\top A_{t-1}^{-1} \boldsymbol{\theta}_t$. With these definitions it easy to see that the conservative version of the algorithm corresponds directly to SOP. The aggressive version corresponds to AROW, with a minor difference. In fact, the prediction of the algorithm in Fig. 1 specialized in this case is $y_t \boldsymbol{w}_t^\top \boldsymbol{x}_t = m_t \frac{r}{r+\chi_t}$, on the other hand AROW predicts with $m_t$. The sign of the predictions is the same, but here the aggressive version is updating when $m_t \frac{r}{r+\chi_t} \le 1$, while AROW updates if $m_t \le 1$.

To derive the bound, observe that using Woodbury matrix identity we have $f_t^*(\theta_t) - f_{t-1}^*(\theta_t) = -\frac{(\boldsymbol{x}_t^\top A_{t-1}^{-1}\boldsymbol{\theta}_t)^2}{2(r+\boldsymbol{x}_t^\top A_{t-1}^{-1}\boldsymbol{x}_t)} = -\frac{m_t^2}{2(r+\chi_t)}$. Using the second bound in Corollary 2, and setting $\eta_t = 1$ we have

$$M + U \le L + \sqrt{\boldsymbol{u}^\top A_T \boldsymbol{u}} \sqrt{\sum_{t\in\mathcal{M}\cup\mathcal{U}} \left( \boldsymbol{x}_t^\top A_t^{-1}\boldsymbol{x}_t + 2y_t \boldsymbol{w}_t^\top \boldsymbol{x}_t - \frac{m_t^2}{r+\chi_t} \right)}$$

$$\le L + \sqrt{\|\boldsymbol{u}\|^2 + \frac{1}{r}\sum_{t\in\mathcal{M}\cup\mathcal{U}}(\boldsymbol{u}^\top \boldsymbol{x}_t)^2} \sqrt{r\log(\det(A_T)) + \sum_{t\in\mathcal{M}\cup\mathcal{U}} \left( 2y_t \boldsymbol{w}_t^\top \boldsymbol{x}_t - \frac{m_t^2}{r+\chi_t} \right)}$$

$$\le L + \sqrt{r\|\boldsymbol{u}\|^2 + \sum_{t\in\mathcal{M}\cup\mathcal{U}}(\boldsymbol{u}^\top \boldsymbol{x}_t)^2} \sqrt{\log(\det(A_T)) + \sum_{t\in\mathcal{M}\cup\mathcal{U}} \frac{m_t(2r - m_t)}{r(r+\chi_t)}} \ .$$

This bound recovers the SOP's one in the conservative case, and improves slightly the one of AROW for the aggressive case. It would be possible to improve the AROW bound even more, setting $\eta_t$ to a value different from 1 in margin error rounds. We leave the details for a longer version of this paper.

## 4.3 Diagonal updates for AROW

Both CW and AROW has an efficient version that use diagonal matrices instead of full ones. In this case the complexity of the algorithm becomes linear in dimension. Here we prove a mistake bound for the diagonal version of AROW, using Corollary 2. We denote $D_t = \text{diag}\{A_t\}$, where $A_t$ is defined as in SOP and AROW, and $f_t(\boldsymbol{x}) = \frac{1}{2}\boldsymbol{x}^\top D_t \boldsymbol{x}$. Setting $\eta_t = 1$, and using the second bound in Corollary 2 and Lemma 12 in [9], we have[1]

$$M + U \le \sum_{t\in\mathcal{M}\cup\mathcal{U}} \ell_t(\boldsymbol{u}) + \sqrt{\boldsymbol{u}^T D_T \boldsymbol{u} \left( r\sum_{i=1}^d \log\left( \frac{\sum_{t\in\mathcal{M}\cup\mathcal{U}} \boldsymbol{x}_{t,i}^2}{r} + 1 \right) + 2U \right)}$$

$$= \sum_{t\in\mathcal{M}\cup\mathcal{U}} \ell_t(\boldsymbol{u}) + \sqrt{\|\boldsymbol{u}\|^2 + \frac{1}{r}\sum_{i=1}^d \boldsymbol{u}_i^2 \sum_{t\in\mathcal{M}\cup\mathcal{U}} \boldsymbol{x}_{t,i}^2} \sqrt{r\sum_{i=1}^d \log\left( \frac{\sum_{t\in\mathcal{M}\cup\mathcal{U}} \boldsymbol{x}_{t,i}^2}{r} + 1 \right) + 2U} \ .$$

The presence of a mistake bound allows us to theoretically analyze the cases where this algorithm could be advantageous respect to a simple Perceptron. In particular, for NLP data the features are binary and it is often the case that most of the features are zero most of the time. On the other hand,

these "rare" features are usually the most informative ones (e.g. [8]). Fig. 2 shows the number of times each feature (word) appears in two sentiment datasets vs the word rank. Clearly there are few very frequent words and many rate words. These exact properties were used to originally derive the CW algorithm. Our analysis justifies this derivation. Concretely, the above considerations leads us to think that the optimal hyperplane $\boldsymbol{u}$ will be such that

$$\sum_{i=1}^{d} \boldsymbol{u}_i^2 \sum_{t\in\mathcal{M}\cup\mathcal{U}} \boldsymbol{x}_{t,i}^2 \approx \sum_{i\in\mathcal{I}} \boldsymbol{u}_i^2 \sum_{t\in\mathcal{M}\cup\mathcal{U}} \boldsymbol{x}_{t,i}^2 \leq \sum_{i\in\mathcal{I}} \boldsymbol{u}_i^2 s \approx s\|\boldsymbol{u}\|^2$$

where $\mathcal{I}$ is the set of the informative and rare features and $s$ is the maximum number of times these features appear in the sequence. In general each time that $\sum_{i=1}^{d} \boldsymbol{u}_i^2 \sum_{t\in\mathcal{M}\cup\mathcal{U}} \boldsymbol{x}_{t,i}^2 \leq s\|\boldsymbol{u}\|^2$ with $s$ small enough, it is possible to show that, with an optimal tuning of $r$, this bound is better of the Perceptron's one. In particular, using a proof similar to the one in [1], in the conservative version of this algorithm, it is enough to have $s < \frac{MR^2}{2d}$, and to set $r = \frac{sMR^2}{MR^2-2sd}$.

## 5    A New Adaptive Second Order Algorithm

We now introduce a new algorithm with an update rule that interpolates from adaptive-second-order-information to fixed-second-order-information. We start from the first bound in Corollary 2. We set $f_t(\boldsymbol{x}) = \frac{1}{2}\boldsymbol{x}^\top A_t \boldsymbol{x}$, where $A_t = A_{t-1} + \frac{\boldsymbol{x}_t \boldsymbol{x}_t^\top}{r_t}$, and $A_0 = I$. This is similar to the regularization used in AROW and SOP, but here we have $r_t > 0$ changing over time. Again, denote $\chi_t = \boldsymbol{x}_t^\top A_{t-1}^{-1} \boldsymbol{x}_t$, and set $\eta_t = 1$. With this choices, we obtain the bound

$$M + U \leq \sum_{t\in\mathcal{M}\cup\mathcal{U}} \ell_t(\boldsymbol{u}) + \frac{\lambda\|\boldsymbol{u}\|^2}{2} + \sum_{t\in\mathcal{M}\cup\mathcal{U}} \left( \frac{\lambda(\boldsymbol{u}^\top \boldsymbol{x}_t)^2}{2r_t} + \frac{\chi_t r_t}{2\lambda(r_t + \chi_t)} - \frac{m_t(2r_t - m_t)}{2\lambda(r_t + \chi_t)} \right),$$

that holds for any $\lambda > 0$ and any choice of $r_t > 0$. We would like to choose $r_t$ at each step to minimize the bound, in particular to have a small value of the sum $\frac{\lambda(\boldsymbol{u}^\top \boldsymbol{x}_t)^2}{r_t} + \frac{\chi_t r_t}{\lambda(r_t + \chi_t)}$. Although we do not know the values of $(\boldsymbol{u}^\top \boldsymbol{x}_t)^2$ and $\lambda$, still we can have a good trade-off setting $r_t = \frac{\chi_t}{b\chi_t - 1}$ when $\chi_t \geq \frac{1}{b}$ and $r_t = +\infty$ otherwise. Here $b$ is a parameter. With this choice we have that $\frac{\chi_t r_t}{r_t + \chi_t} = \frac{1}{b}$, and $\frac{(\boldsymbol{u}^\top \boldsymbol{x}_t)^2}{r_t} = \frac{\chi_t(\boldsymbol{u}^\top \boldsymbol{x}_t)^2 b}{r_t + \chi_t}$, when $\chi_t \geq \frac{1}{b}$. Hence we have

$$M + U - \frac{\lambda\|\boldsymbol{u}\|^2}{2} - \sum_{t\in\mathcal{M}\cup\mathcal{U}} \ell_t(\boldsymbol{u})$$

$$\leq \sum_{t:b\chi_t>1} \left( \frac{\lambda b\chi_t(\boldsymbol{u}^\top \boldsymbol{x}_t)^2}{2(r_t + \chi_t)} + \frac{1}{2\lambda b} \right) + \frac{1}{2\lambda} \sum_{t:b\chi_t\leq 1} \chi_t - \sum_{t\in\mathcal{M}\cup\mathcal{U}} \frac{m_t(2r_t - m_t)}{2\lambda(r_t + \chi_t)}$$

$$\leq \lambda b \sum_{t:b\chi_t>1} \frac{\chi_t\|\boldsymbol{u}\|^2 R^2}{2(r_t + \chi_t)} + \frac{1}{2\lambda} \sum_{t\in\mathcal{M}\cup\mathcal{U}} \min\left(\frac{1}{b}, \chi_t\right) - \sum_{t\in\mathcal{M}\cup\mathcal{U}} \frac{m_t(2r_t - m_t)}{2\lambda(r_t + \chi_t)}$$

$$\leq \frac{1}{2}\lambda bR^2\|\boldsymbol{u}\|^2 \log\det(A_T) + \frac{1}{2\lambda} \sum_{t\in\mathcal{M}\cup\mathcal{U}} \min\left(\frac{1}{b}, \chi_t\right) - \sum_{t\in\mathcal{M}\cup\mathcal{U}} \frac{m_t(2r_t - m_t)}{2\lambda(r_t + \chi_t)},$$

where in the last inequality we used an extension of Lemma 4 in [5] to varying values of $r_t$. Tuning $\lambda$ we have

$$M + U \leq \sum_{t\in\mathcal{M}\cup\mathcal{U}} \ell_t(\boldsymbol{u}) + \|\boldsymbol{u}\| R \sqrt{\frac{1}{bR^2} + \log\det(A_T)} \sqrt{\sum_{t\in\mathcal{M}\cup\mathcal{U}} \left( \min(1, b\chi_t) - \frac{bm_t(2r_t - m_t)}{r_t + \chi_t} \right)}.$$

This algorithm interpolates between a second order algorithm with adaptive second order information, like AROW, and one with a fixed second order information. Even the bound is in between these two worlds. In particular the matrix $A_t$ is updated only if $\chi_t \geq \frac{1}{b}$, preventing its eigenvalues from growing too much, as in AROW/SOP. We thus call this algorithm NAROW, since its is a new adaptive algorithm, which narrows the range of possible eigenvalues of the matrix $A_t$. We illustrate empirically its properties in the next section.

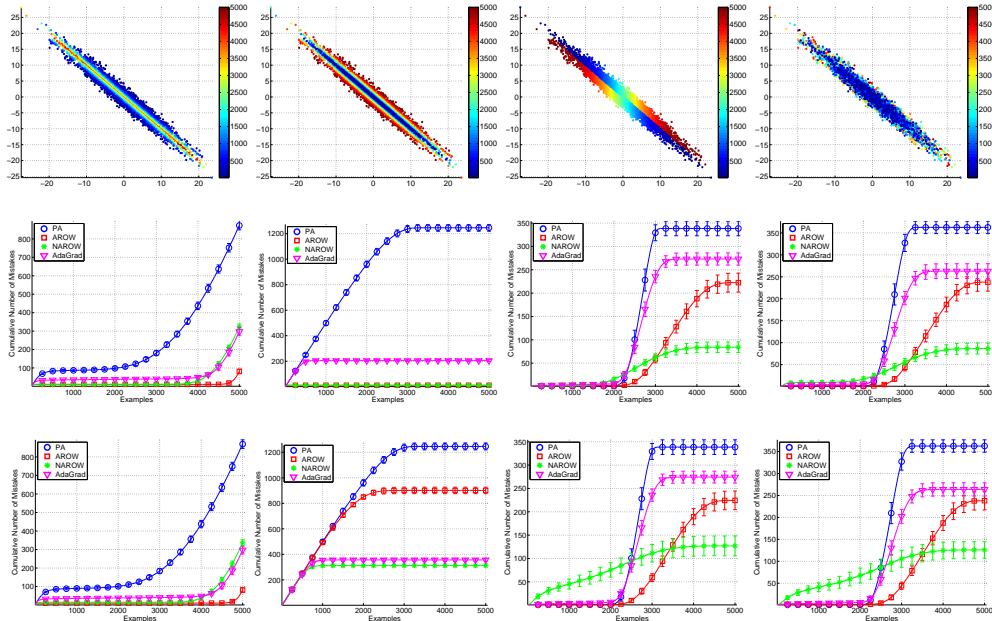

Figure 3: Top: Four sequences used for training, the colors represents the ordering in the sequence from blue to yellow, to red. Middle: cumulative number of mistakes of four algorithms on data with no labels noise. Bottom: results when training using data with 10% label-noise.

## 6   Experiments

We illustrate the characteristics of our algorithm NAROW using a synthetic data generated in a similar manner of previous work [4]. We repeat its properties for completeness. We generated $5,000$ points in $\mathbb{R}^{20}$ where the first two coordinates were drawn from a $45°$ rotated Gaussian distribution with standard deviation 1 and 10. The remaining 18 coordinates were drawn from independent Gaussian distributions $\mathcal{N}(0, 8.5)$. Each point's label depended on the first two coordinates using a separator parallel to the long axis of the ellipsoid, yielding a linearly separable set. Finally, we ordered the training set in four different ways: from easy examples to hard examples (measured by the signed distance to the separating-hyperplane), from hard examples to easy examples, ordered by their signed value of the first feature, and by the signed value of the third (noisy) feature - that is by $x_i \times y$ for $i = 1$ and $i = 3$ - respectively. An illustration of these ordering appears in the top row of Fig. 3, the colors code the ordering of points from blue via yellow to red (last points). We evaluated four algorithms: version I of the passive-aggressive (PA-I) algorithm [3], AROW [5], AdaGrad [9] and NAROW. All algorithms, except AdaGrad, have one parameter to be tuned, while AdaGrad has two. These parameters were chosen on a single random set, and the plots summarizes the results averaged over 100 repetitions.

The second row of Fig. 3 summarizes the cumulative number of mistakes averaged over 100 repetitions and the third row shows the cumulative number of mistakes where $10\%$ artificial label noise was used. (Mistakes are counted using the unnoisy labels.)

Focusing on the left plot, we observe that all the second order algorithms outperform the single first order algorithm - PA-I. All algorithms make few mistakes when receiving the first half of the data - the easy examples. Then all algorithms start to make more mistakes - PA-I the most, then AdaGrad and closely following NAROW, and AROW the least. In other words, AROW was able to converge faster to the target separating hyperplane just using "easy" examples which are far from the separating hyperplane, then NAROW and AdaGrad, with PA-I being the worst in this aspect.

The second plot from the left, showing the results for ordering the examples from hard to easy. All algorithms follow a general trend of making mistakes in a linear rate and then stop making mistakes when the data is easy and there are many possible classifiers that can predict correctly. Clearly,

AROW and NAROW stop making mistakes first, then AdaGrad and PA-I last. A similar trend can be found in the noisy dataset, with each algorithm making relatively more mistakes.

The third and fourth columns tell a similar story, although the plots in the third column summarize results when the instances are ordered using the first feature (which is informative with the second) and the plots in the fourth column summarize when the instances are ordered using the third uninformative feature. In both cases, all algorithms do not make many mistakes in the beginning, then at some point, close to the middle of the input sequence, they start making many mistakes for a while, and then they converge. In terms of total performance: PA-I makes more mistakes, then AdaGrad, AROW and NAROW. However, NAROW starts to make many mistakes before the other algorithms and takes more "examples" to converge until it stopped making mistakes. This phenomena is further shown in the bottom plots where label noise is injected.

We hypothesize that this relation is due to the fact that NAROW does not let the eigenvalues of the matrix $A$ to grow unbounded. Since its inverse is proportional to the effective learning rate, it means that it does not allow the learning rate to drop too low as opposed to AROW and even to some extent AdaGrad.

## 7 Conclusion

We presented a framework for online convex classification, specializing it for particular losses, as the hinge loss. This general tool allows to design theoretical motivated online classification algorithms and to prove their relative mistake bound. In particular it supports the analysis of aggressive updates. Our framework also provided a missing bound for AROW for diagonal matrices. We have shown its utility proving better bounds for known online algorithms, and proposing a new algorithm, called NAROW. This is a hybrid between adaptive second order algorithms, like AROW and SOP, and a static second order one. We have validated it using synthetic datasets, showing its robustness to the malicious orderings of the sample, comparing it with other state-of-art algorithms. Future work will focus on exploring the new possibilities offered by our framework and on testing NAROW on real world data.

**Acknowledgments** We thank Nicolò Cesa-Bianchi for his helpful comments. Francesco Orabona was sponsored by the PASCAL2 NoE under EC grant no. 216886. Koby Crammer is a Horev Fellow, supported by the Taub Foundations. This work was also supported by the German-Israeli Foundation grant GIF-2209-1912.

## A Appendix

*Proof of Lemma 1.* Define by $f_t^*$ the Fenchel dual of $f_t$, and $\Delta_t = f_t^*(\boldsymbol{\theta}_{t+1}) - f_{t-1}^*(\boldsymbol{\theta}_t)$. We have $\sum_{t=1}^{T} \Delta_t = f_T^*(\boldsymbol{\theta}_{T+1}) - f_0^*(\boldsymbol{\theta}_1) = f_T^*(\boldsymbol{\theta}_{T+1})$. Moreover we have that $\Delta_t = f_t^*(\boldsymbol{\theta}_{t+1}) - f_t^*(\boldsymbol{\theta}_t) + f_t^*(\boldsymbol{\theta}_t) - f_{t-1}^*(\boldsymbol{\theta}_t) \leq f_t^*(\boldsymbol{\theta}_t) - f_{t-1}^*(\boldsymbol{\theta}_t) - \eta_t \boldsymbol{z}_t^\top \nabla f_t^*(\boldsymbol{\theta}_t) + \frac{\eta_t^2}{2\beta_t} \|\boldsymbol{z}_t\|_{f_t^*}^2$, where we used Theorem 6 in [13]. Moreover using the Fenchel-Young inequality, we have that $\frac{1}{\lambda} \sum_{t=1}^{T} \Delta_t = \frac{1}{\lambda} f_T^*(\boldsymbol{\theta}_{T+1}) \geq \boldsymbol{u}^\top \boldsymbol{\theta}_{T+1} - \frac{1}{\lambda} f_T(\lambda \boldsymbol{u}) = -\sum_{t=1}^{T} \eta_t \boldsymbol{u}^\top \boldsymbol{z}_t - \frac{1}{\lambda} f_T(\lambda \boldsymbol{u})$. Hence putting all togheter we have

$$-\sum_{t=1}^{T} \eta_t \boldsymbol{u}^\top \boldsymbol{z}_t - \frac{1}{\lambda} f_T(\lambda \boldsymbol{u}) \leq \frac{1}{\lambda} \sum_{t=1}^{T} \Delta_t \leq \frac{1}{\lambda} \sum_{t=1}^{T} (f_t^*(\boldsymbol{\theta}_t) - f_{t-1}^*(\boldsymbol{\theta}_t) - \eta_t \boldsymbol{w}_t^\top \boldsymbol{z}_t + \frac{\eta_t^2}{2\beta_t} \|\boldsymbol{z}_t\|_{f_t^*}^2),$$

where we used the definition of $\boldsymbol{w}_t$ in Algorithm 1. □

*Proof of Corollary 1.* By convexity, $\ell(\boldsymbol{w}_t, \boldsymbol{x}_t, y_t) - \ell(\boldsymbol{u}, \boldsymbol{x}_t, y_t) \leq \boldsymbol{z}_t^\top (\boldsymbol{w}_t - \boldsymbol{u})$, so setting $\lambda = 1$ in Lemma 1 we have the stated bound. For the additional statement, using Lemma 12 in [16] and $f_t(\boldsymbol{x}) \leq f_{t+1}(\boldsymbol{x})$ we have that $f_t^*(\boldsymbol{x}) \geq f_{t+1}^*(\boldsymbol{x})$, so $B \leq 0$. The additional statement on $B$ is proved using Lemma 12 in [16]. Using it, we have that $f_t(\boldsymbol{x}) \leq f_{t+1}(\boldsymbol{x})$ implies that $f_t^*(\boldsymbol{x}) \geq f_{t+1}^*(\boldsymbol{x})$, so we have that $B \leq 0$. □

*Proof of Corollary 2.* Lemma 1, the condition on the loss (2), and the hypothesis on $f_T$ gives us

$$\sum_{t=1}^{T} \eta_t (1 - \ell_t(\boldsymbol{u})) \leq -\sum_{t=1}^{T} \eta_t \boldsymbol{u}^\top \boldsymbol{z}_t \leq \lambda f_T(\boldsymbol{u}) + \frac{1}{\lambda} \sum_{t=1}^{T} \left( \frac{\eta_t^2 \|\boldsymbol{z}_t\|_{f_t^*}^2}{2\beta_t} + B - \eta_t \boldsymbol{z}_t^\top \boldsymbol{w}_t \right).$$

Note that $\lambda$ is free, so choosing its optimal value we get the second bound. □

## Footnotes

[1] We did not optimize the constant multiplying $U$ in the bound.

# References

[1] N. Cesa-Bianchi, A. Conconi, and C. Gentile. A second-order Perceptron algorithm. *SIAM Journal on Computing*, 34(3):640–668, 2005.

[2] N. Cesa-Bianchi and G. Lugosi. *Prediction, learning, and games*. Cambridge University Press, 2006.

[3] K. Crammer, O. Dekel, J. Keshet, S. Shalev-Shwartz, and Y. Singer. Online passive-aggressive algorithms. *Journal of Machine Learning Research*, 7:551–585, 2006.

[4] K. Crammer, M. Dredze, and F. Pereira. Exact Convex Confidence-Weighted learning. *Advances in Neural Information Processing Systems*, 22, 2008.

[5] K. Crammer, A. Kulesza, and M. Dredze. Adaptive regularization of weight vectors. *Advances in Neural Information Processing Systems*, 23, 2009.

[6] K. Crammer and Y. Singer. Ultraconservative online algorithms for multiclass problems. *Journal of Machine Learning Research*, 3:951–991, 2003.

[7] N. Cristianini and J. Shawe-Taylor. *An Introduction to Support Vector Machines and Other Kernel-Based Learning Methods*. Cambridge University Press, 2000.

[8] M. Dredze, K. Crammer, and F. Pereira. Online Confidence-Weighted learning. *Proceedings of the 25th International Conference on Machine Learning*, 2008.

[9] J. Duchi, E. Hazan, and Y. Singer. Adaptive subgradient methods for online learning and stochastic optimization. Technical Report 2010-24, UC Berkeley Electrical Engineering and Computer Science, 2010. Available at `http://cs.berkeley.edu/˜jduchi/ projects/DuchiHaSi10.pdf`.

[10] Y. Freund and R. E. Schapire. Large margin classification using the Perceptron algorithm. *Machine Learning*, pages 277–296, 1999.

[11] C. Gentile. The robustness of the p-norm algorithms. *Machine Learning*, 53(3):265–299, 2003.

[12] E. Hazan and S. Kale. Extracting certainty from uncertainty: Regret bounded by variation in costs. In *Proc. of the 21st Conference on Learning Theory*, 2008.

[13] S. Kakade, S. Shalev-Shwartz, and A. Tewari. On the duality of strong convexity and strong smoothness: Learning applications and matrix regularization. Technical report, TTI, 2009. http://www.cs.huji.ac.il/ shais/papers/KakadeShalevTewari09.pdf.

[14] J. Kivinen, A. Smola, and R. Williamson. Online learning with kernels. *IEEE Trans. on Signal Processing*, 52(8):2165–2176, 2004.

[15] A. Rakhlin and A. Tewari. Lecture notes on online learning. Technical report, 2008. Available at `http://www-stat.wharton.upenn.edu/˜rakhlin/papers/online_ learning.pdf`.

[16] S. Shalev-Shwartz. Online learning: Theory, algorithms, and applications. Technical report, The Hebrew University, 2007. PhD thesis.

[17] S. Shalev-Shwartz and Y. Singer. A primal-dual perspective of online learning algorithms. *Machine Learning Journal*, 2007.

[18] L. Xiao. Dual averaging method for regularized stochastic learning and online optimization. In *Advances in Neural Information Processing Systems 22*, pages 2116–2124. 2009.

